# Sketch-Based Linear Value Function Approximation

**Marc G. Bellemare**
University of Alberta
mg17@cs.ualberta.ca

**Joel Veness**
University of Alberta
veness@cs.ualberta.ca

**Michael Bowling**
University of Alberta
bowling@cs.ualberta.ca

## Abstract

Hashing is a common method to reduce large, potentially infinite feature vectors to a fixed-size table. In reinforcement learning, hashing is often used in conjunction with tile coding to represent states in continuous spaces. Hashing is also a promising approach to value function approximation in large discrete domains such as Go and Hearts, where feature vectors can be constructed by exhaustively combining a set of atomic features. Unfortunately, the typical use of hashing in value function approximation results in biased value estimates due to the possibility of collisions. Recent work in data stream summaries has led to the development of the tug-of-war sketch, an unbiased estimator for approximating inner products. Our work investigates the application of this new data structure to linear value function approximation. Although in the reinforcement learning setting the use of the tug-of-war sketch leads to biased value estimates, we show that this bias can be orders of magnitude less than that of standard hashing. We provide empirical results on two RL benchmark domains and fifty-five Atari 2600 games to highlight the superior learning performance obtained when using tug-of-war hashing.

## 1   Introduction

Recent value-based reinforcement learning applications have shown the benefit of exhaustively generating features, both in discrete and continuous state domains. In discrete domains, exhaustive feature generation combines atomic features into logical predicates. In the game of Go, Silver et al. [19] showed that good features could be generated by enumerating all stone patterns up to a certain size. Sturtevant and White [21] similarly obtained promising reinforcement learning results using a feature generation method that enumerated all 2, 3 and 4-wise combinations of a set of 60 atomic features. In continuous-state RL domains, tile coding [23] is a canonical example of exhaustive feature generation; tile coding has been successfully applied to benchmark domains [22], to learn to play keepaway soccer [20], in multiagent robot learning [4], to train bipedal robots to walk [18, 24] and to learn mixed strategies in the game of Goofspiel [3].

Exhaustive feature generation, however, can result in feature vectors that are too large to be represented in memory, especially when applied to continuous spaces. Although such feature vectors are too large to be represented explicitly, in many domains of interest they are also sparse. For example, most stone patterns are absent from any particular Go position. Given a fixed memory budget, the standard approach is to hash features into a fixed-size table, with collisions implicitly handled by the learning algorithm; all but one of the applications discussed above use some form of hashing.

With respect to its typical use for linear value function approximation, hashing lacks theoretical guarantees. In order to improve on the basic hashing idea, we turn to sketches: state-of-the-art methods for approximately storing large vectors [6]. Our goal is to show that one such sketch, the *tug-of-war* sketch [7], is particularly well-suited for linear value function approximation. Our work is related to recent developments on the use of random projections in reinforcement learning [11] and least-squares regression [16, 10]. Hashing, however, possesses a computational advantage over traditional random projections: each feature is hashed exactly once. In comparison, even sparse

random projection methods [1, 14] carry a per-feature cost that increases with the size of the reduced space. Tug-of-war hashing seeks to reconcile the computational efficiency that makes hashing a practical method for linear value function approximation on large feature spaces, while preserving the theoretical appeal of random projection methods.

A natural concern when using hashing in RL is that hash collisions irremediably degrade learning. In this paper we argue that tug-of-war hashing addresses this concern by providing us with a low-error approximation of large feature vectors at a fraction of the memory cost. To quote Sutton and Barto [23], *"Hashing frees us from the curse of dimensionality in the sense that memory requirements need not be exponential in the number of dimensions, but need merely match the real demands of the task."*

## 2   Background

We consider the reinforcement learning framework of Sutton and Barto [23]. An MDP $\mathcal{M}$ is a tuple $\langle \mathcal{S}, \mathcal{A}, P, R, \gamma \rangle$, where $\mathcal{S}$ is the set of states, $\mathcal{A}$ is the set of actions, $P : \mathcal{S} \times \mathcal{A} \times \mathcal{S} \rightarrow [0, 1]$ is the transition probability function, $R : \mathcal{S} \times \mathcal{A} \rightarrow \mathbb{R}$ is the reward function and $\gamma \in [0, 1]$ is the discount factor. At time step $t$ the agent observes state $s_t \in \mathcal{S}$, selects an action $a_t \in \mathcal{A}$ and receives a reward $r_t := R(s_t, a_t)$. The agent then observes the new state $s_{t+1}$ distributed according to $P(\cdot | s_t, a_t)$. From state $s_t$, the agent's goal is to maximize the expected discounted sum of future rewards $\mathbb{E}\left[ \sum_{i=0}^{\infty} \gamma^i R(s_{t+i}, a_{t+i}) \right]$. A typical approach is to learn state-action values $Q^\pi(s, a)$, where the stationary policy $\pi : \mathcal{S} \times \mathcal{A} \rightarrow [0, 1]$ represents the agent's behaviour. $Q^\pi(s, a)$ is recursively defined as:

$$Q^\pi(s, a) := R(s, a) + \gamma \mathbb{E}_{s' \sim P(\cdot | s, a)} \left[ \sum_{a' \in \mathcal{A}} \pi(a' | s') Q^\pi(s', a') \right] \tag{1}$$

A special case of this equation is the *optimal value function* $Q^*(s, a) := R(s, a) + \gamma \mathbb{E}_{s'} \left[ \max_{a'} Q^*(s', a') \right]$. The optimal value function corresponds to the value under an optimal policy $\pi^*$. For a fixed $\pi$, The SARSA($\lambda$) algorithm [23] learns $Q^\pi$ from sample transitions $(s_t, a_t, r_t, s_{t+1}, a_{t+1})$. In domains where $\mathcal{S}$ is large (or infinite), learning $Q^\pi$ exactly is impractical and one must rely on *value function approximation*. A common value function approximation scheme in reinforcement learning is linear approximation. Given $\phi : \mathcal{S} \times \mathcal{A} \rightarrow \mathbb{R}^n$ mapping state-action pairs to feature vectors, we represent $Q^\pi$ with the linear approximation $Q_t(s, a) := \theta_t \cdot \phi(s, a)$, where $\theta_t \in \mathbb{R}^n$ is a weight vector. The gradient descent SARSA($\lambda$) update is defined as:

$$\begin{aligned} \delta_t &\leftarrow r_t + \gamma \theta_t \cdot \phi(s_{t+1}, a_{t+1}) - \theta_t \cdot \phi(s_t, a_t) \\ e_t &\leftarrow \gamma \lambda e_{t-1} + \phi(s_t, a_t) \\ \theta_{t+1} &\leftarrow \theta_t + \alpha \delta_t e_t \, , \end{aligned} \tag{2}$$

where $\alpha \in [0, 1]$ is a step-size parameter and $\lambda \in [0, 1]$ controls the degree to which changes in the value function are propagated back in time. Throughout the rest of this paper $Q^\pi(s, a)$ refers to the exact value function computed from Equation 1 and we use $Q_t(s, a)$ to refer to the linear approximation $\theta_t \cdot \phi(s, a)$; "gradient descent SARSA($\lambda$) with linear approximation" is always implied when referring to SARSA($\lambda$). We call $\phi(s, a)$ the *full feature vector* and $Q_t(s, a)$ the *full-vector value function*.

Asymptotically, SARSA($\lambda$) is guaranteed to find the best solution within the span of $\phi(s, a)$, up to a multiplicative constant that depends on $\lambda$ [25]. If we let $\Phi \in \mathbb{R}^{|\mathcal{S}||\mathcal{A}| \times n}$ denote the matrix of full feature vectors $\phi(s, a)$, and let $\mu : \mathcal{S} \times \mathcal{A} \rightarrow [0, 1]$ denote the steady state distribution over state-action pairs induced by $\pi$ and $P$ then, under mild assumptions, we can guarantee the existence and uniqueness of $\mu$. We denote by $\langle \cdot, \cdot \rangle_\mu$ the inner product induced by $\mu$, i.e. $\langle x, y \rangle_\mu := x^T D y$, where $x, y \in \mathbb{R}^{|\mathcal{S}||\mathcal{A}|}$ and $D \in \mathbb{R}^{|\mathcal{S}||\mathcal{A}| \times |\mathcal{S}||\mathcal{A}|}$ is a diagonal matrix with entries $\mu(s, a)$. The norm $\| \cdot \|_\mu$ is defined as $\sqrt{\langle \cdot, \cdot \rangle_\mu}$. We assume the following: 1) $\mathcal{S}$ and $\mathcal{A}$ are finite, 2) the Markov chain induced by $\pi$ and $P$ is irreducible and aperiodic, and 3) $\Phi$ has full rank. The following theorem bounds the error of SARSA($\lambda$):

**Theorem 1** (Restated from Tsitsiklis and Van Roy [25]). *Let* $\mathcal{M} = \langle \mathcal{S}, \mathcal{A}, P, R, \gamma \rangle$ *be an MDP and* $\pi : \mathcal{S} \times \mathcal{A} \rightarrow [0, 1]$ *be a policy. Denote by* $\Phi \in \mathbb{R}^{|\mathcal{S}||\mathcal{A}| \times n}$ *the matrix of full feature vectors and*

*by $\mu$ the stationary distribution on $(\mathcal{S}, \mathcal{A})$ induced by $\pi$ and $P$. Under assumptions 1-3), SARSA($\lambda$) converges to a unique $\theta^\pi \in \mathbb{R}^n$ with probability one and*

$$\left\| \Phi \theta^\pi - Q^\pi \right\|_\mu \leq \frac{1 - \lambda \gamma}{1 - \gamma} \left\| \mathbf{\Pi} Q^\pi - Q^\pi \right\|_\mu ,$$

*where $Q^\pi \in \mathbb{R}^{|\mathcal{S}||\mathcal{A}|}$ is a vector representing the exact solution to Equation 1 and $\mathbf{\Pi} := \Phi(\Phi^T D \Phi)^{-1} \Phi^T D$ is the projection operator.*

Because $\mathbf{\Pi}$ is the projector operator for $\Phi$, for any $\theta$ we have $\left\| \Phi \theta - Q^\pi \right\|_\mu \geq \left\| \mathbf{\Pi} Q^\pi - Q^\pi \right\|_\mu$; Theorem 1 thus implies that SARSA(1) converges to $\theta^\pi = \arg\min_\theta \left\| \Phi \theta - Q^\pi \right\|_\mu$.

## 2.1 Hashing in Reinforcement Learning

As discussed previously, it is often impractical to store the full weight vector $\theta_t$ in memory. A typical example of this is tile coding on continuous-state domains [22], which generates a number of features exponential in the dimensionality of the state space. In such cases, hashing can effectively be used to approximate $Q^\pi(s, a)$ using a fixed memory budget. Let $h$ be a hash function $h : \{1, \ldots, n\} \to \{1, \ldots, m\}$, mapping full feature vector indices into hash table indices, where $m \ll n$ is the hash table size. We define *standard hashing features* as the feature map $\hat{\phi}(s, a)$ whose $i^{\text{th}}$ component is defined as:

$$\hat{\phi}_i(s, a) := \sum_{j=1}^n \mathbb{I}_{[h(j)=i]} \phi_j(s, a) , \tag{3}$$

where $\phi_j(s, a)$ denotes the $j^{th}$ component of $\phi(s, a)$ and $\mathbb{I}_{[x]}$ denotes the indicator function. We assume that our hash function $h$ is drawn from a universal family: for any $i, j \in \{1, \ldots, n\}, i \neq j$, $\Pr(h(i) = h(j)) \leq \frac{1}{m}$.[1] We define the *standard hashing value function* $\hat{Q}_t(s, a) := \hat{\theta}_t \cdot \hat{\phi}(s, a)$, where $\hat{\theta}_t \in \mathbb{R}^m$ is a weight vector, and $\hat{\phi}(s, a)$ is the *hashed vector*. Because of hashing collisions, the standard hashing value function is a biased estimator of $Q_t(s, a)$, i.e., in general $\mathbb{E}_h[\hat{Q}_t(s, a)] \neq Q_t(s, a)$. For example, consider the extreme case where $m = 1$: all features share the same weight. We return to the issue of the bias introduced by standard hashing in Section 4.1.

## 2.2 Tug-of-War Hashing

The tug-of-war sketch, also known as the Fast-AGMS, was recently introduced as a powerful method for approximating inner products of large vectors [7]. The name "sketch" refers to the data structure's function as a summary of a stream of data. In the canonical sketch setting, we summarize a count vector $\theta \in \mathbb{R}^n$ using a sketch vector $\tilde{\theta} \in \mathbb{R}^m$. At each time step a vector $\phi_t \in \mathbb{R}^n$ is received. The purpose of the sketch vector is to approximate the count vector $\theta_t := \sum_{i=0}^{t-1} \phi_i$. Given two hash functions, $h$ and $\xi : \{1, \ldots, n\} \to \{-1, 1\}$, $\phi_t$ is mapped to a vector $\tilde{\phi}_t$ whose $i^{th}$ component is

$$\tilde{\phi}_{t,i} := \sum_{j=1}^n \mathbb{I}_{[h(j)=i]} \phi_{t,j} \xi(j) \tag{4}$$

The tug-of-war sketch vector is then updated as $\tilde{\theta}_{t+1} \leftarrow \tilde{\theta}_t + \tilde{\phi}_t$. In addition to $h$ being drawn from a universal family of hash functions, $\xi$ is drawn from a four-wise independent family of hash functions: for all sets of four unique indices $\{i_1, i_2, i_3, i_4\}$, $\Pr_\xi(\xi(i_1) = k_1, \xi(i_2) = k_2, \xi(i_3) = k_3, \xi(i_4) = k_4) = \frac{1}{16}$ with $k_1 \ldots k_4 \in \{-1, 1\}$. For an arbitrary $\phi \in \mathbb{R}^n$ and its corresponding tug-of-war vector $\tilde{\phi} \in \mathbb{R}^m$, $\mathbb{E}_{h,\xi}[\tilde{\theta}_t \cdot \tilde{\phi}] = \theta_t \cdot \phi$: the tug-of-war sketch produces unbiased estimates of inner products [7]. This unbiasedness property can be derived as follows. First let $\tilde{\theta}_t = \sum_{i=0}^{t-1} \tilde{\phi}_i$.

Then $\tilde{\theta}_t \cdot \tilde{\phi}_{t'} = \sum_{i=0}^{t-1} \tilde{\phi}_i \cdot \tilde{\phi}_{t'}$ and

$$
\begin{aligned}
\mathbb{E}_{h,\xi}[\tilde{\phi}_i \cdot \tilde{\phi}_{t'}] &= \mathbb{E}_{h,\xi}\left[ \sum_{j_1=1}^{n} \sum_{j_2=1}^{n} \mathbb{I}_{[h(j_1)=h(j_2)]} \phi_{i,j_1} \phi_{t',j_2} \xi(j_1)\xi(j_2) \right] \\
\mathbb{E}_{\xi}[\xi(j_1)\xi(j_2)] &= \left\{ \begin{array}{ll} 1 & \text{if } j_1 = j_2 \\ 0 & \text{otherwise} \end{array} \right. \qquad \text{(by four-wise independence)}
\end{aligned}
$$

The result follows by noting that $\mathbb{I}_{[h(j_1)=h(j_2)]}$ is independent from $\xi(j_1)\xi(j_2)$ given $j_1, j_2$.

## 3  Tug-of-War with Linear Value Function Approximation

We now extend the tug-of-war sketch to the reinforcement learning setting by defining the *tug-of-war hashing features* as $\tilde{\phi} : \mathcal{S} \times \mathcal{A} \to \mathbb{R}^m$ with $\tilde{\phi}_i(s,a) := \sum_{j=1}^{n} \mathbb{I}_{[h(j)=i]} \phi_j(s,a)\xi(j)$. The SARSA($\lambda$) update becomes:

$$
\begin{aligned}
\tilde{\delta}_t &\leftarrow r_t + \gamma \tilde{\theta}_t \cdot \tilde{\phi}(s_{t+1}, a_{t+1}) - \tilde{\theta}_t \cdot \tilde{\phi}(s_t, a_t) \\
\tilde{e}_t &\leftarrow \gamma\lambda \tilde{e}_{t-1} + \tilde{\phi}(s_t, a_t) \\
\tilde{\theta}_{t+1} &\leftarrow \tilde{\theta}_t + \alpha \tilde{\delta}_t \tilde{e}_t.
\end{aligned} \tag{5}
$$

We also define the *tug-of-war value function* $\tilde{Q}_t(s,a) := \tilde{\theta}_t \cdot \tilde{\phi}(s,a)$ with $\tilde{\theta}_t \in \mathbb{R}^m$ and refer to $\tilde{\phi}(s,a)$ as the *tug-of-war vector*.

### 3.1  Value Function Approximation with Tug-of-War Hashing

Intuitively, one might hope that the unbiasedness of the tug-of-war sketch for approximating inner products carries over to the case of linear value function approximation. Unfortunately, this is not the case. However, it is still possible to bound the error of the tug-of-war value function learned with SARSA(1) in terms of the full-vector value function. Our bound relies on interpreting tug-of-war hashing as a special kind of Johnson-Lindenstrauss transform [8].

We define a $\infty$-universal family of hash functions $\mathbb{H}$ such that for any set of indices $i_1, i_2, \ldots, i_l$ $\Pr(h(i_1) = k_1, \ldots, h(i_l) = k_l) \le \frac{1}{|\mathscr{C}|^l}$, where $\mathscr{C} \subset \mathbb{N}$ and $h \in \mathbb{H} : \{1, \ldots, n\} \to \mathscr{C}$.

**Lemma 1** (Dasgupta *et al.* [8], Theorem 2). *Let $h : \{1, \ldots, n\} \to \{1, \ldots, m\}$ and $\xi : \{1, \ldots, n\} \to \{-1, 1\}$ be two independent hash functions chosen uniformly at random from $\infty$-universal families and let $H \in \{0, \pm1\}^{m \times n}$ be a matrix with entries $H_{ij} = \mathbb{I}_{[h(j)=i]}\xi(j)$. Let $\epsilon < 1$, $\delta < \frac{1}{10}$, $m = \frac{12}{\epsilon^2} \log\left(\frac{1}{\delta}\right)$ and $c = \frac{16}{\epsilon} \log\left(\frac{1}{\delta}\right) \log^2\left(\frac{m}{\delta}\right)$. For any given vector $x \in \mathbb{R}^n$ such that $\|x\|_\infty \le \frac{1}{\sqrt{c}}$, with probability $1 - 3\delta$, $H$ satisfies the following property:*

$$
(1 - \epsilon) \|x\|_2^2 \le \|Hx\|_2^2 \le (1 + \epsilon) \|x\|_2^2.
$$

Lemma 1 states that, under certain conditions on the input vector $x$, tug-of-war hashing approximately preserves the norm of $x$. When $\delta$ and $\epsilon$ are constant, the requirement on $\|x\|_\infty$ can be waived by applying Theorem 1 to the normalized vector $u = \frac{x}{\|x\|_2 \sqrt{c}}$. A clear discussion on hashing as a Johnson-Lindenstrauss transform can be found in the work of Kane and Nelson [13], who also improve Lemma 1 and extend it to the case where the family of hash functions is $k$-universal rather than $\infty$-universal.

**Lemma 2** (Based on Maillard and Munos [16], Proposition 1). *Let $x_1 \ldots x_K$ and $y$ be vectors in $\mathbb{R}^n$. Let $H \in \{0, \pm1\}^{m \times n}$, $\epsilon$, $\delta$ and $m$ be defined as in Lemma 1. With probability at least $1 - 6K\delta$, for all $k \in \{1, \ldots, K\}$,*

$$
x_k \cdot y - \epsilon \|x_k\|_2 \|y\|_2 \le Hx_k \cdot Hy \le x_k \cdot y + \epsilon \|x_k\|_2 \|y\|_2.
$$

*Proof (Sketch).* The proof follows the steps of Maillard and Munos [16]. Given two unit vectors $u, v \in \mathbb{R}^n$, we can relate $(Hu) \cdot (Hv)$ to $\|Hu + Hv\|_2^2$ and $\|Hu - Hv\|_2^2$ using the parallelogram law. We then apply Lemma 1 to bound both sides of each squared norm and substitute $x_k$ for $u$ and $y$ for $w$ to bound $Hx_k \cdot Hy$. Applying the union bound yields the desired statement. $\square$

We are now in a position to bound the asymptotic error of SARSA(1) with tug-of-war hashing. Given hash functions $h$ and $\xi$ defined as per Lemma 1, we denote by $H \in \mathbb{R}^{m \times n}$ the matrix whose entries are $H_{ij} := \mathbb{I}_{[h(j)=i]}\xi(j)$, such that $\tilde{\phi}(s,a) = H\phi(s,a)$. We also denote by $\tilde{\Phi} := \Phi H^T$ the matrix of tug-of-war vectors. We again assume that 1) $\mathcal{S}$ and $\mathcal{A}$ are finite, that 2) $\pi$ and $P$ induce an irreducible, aperiodic Markov chain and that 3) $\Phi$ has full rank. For simplicity of argument, we also assume that 4) $\tilde{\Phi} := \Phi H^T$ has full rank; when $\tilde{\Phi}$ is rank-deficient, SARSA(1) converges to a set of solutions $\tilde{\Theta}^\pi$ satisfying the bound of Theorem 2, rather than to a unique $\tilde{\theta}^\pi$.

**Theorem 2.** *Let $\mathcal{M} = \langle \mathcal{S}, \mathcal{A}, P, R, \gamma \rangle$ be an MDP and $\pi : \mathcal{S} \times \mathcal{A} \to [0,1]$ be a policy. Let $\Phi \in \mathbb{R}^{|\mathcal{S}||\mathcal{A}| \times n}$ be the matrix of full feature vectors and $\tilde{\Phi} \in \mathbb{R}^{|\mathcal{S}||\mathcal{A}| \times m}$ be the matrix of tug-of-war vectors. Denote by $\mu$ the stationary distribution on $(\mathcal{S}, \mathcal{A})$ induced by $\pi$ and $P$. Let $\epsilon < 1$, $\delta < 1$, $\delta' = \frac{\delta}{6|\mathcal{S}||\mathcal{A}|}$ and $m \geq \frac{12}{\epsilon^2} \log \frac{1}{\delta'}$. Under assumptions 1-4), gradient-descent SARSA(1) with tug-of-war hashing converges to a unique $\tilde{\theta}^\pi \in \mathbb{R}^m$ and with probability at least $1 - \delta$*

$$\left\| \tilde{\Phi}\tilde{\theta}^\pi - Q^\pi \right\|_\mu \leq \left\| \Phi\theta^\pi - Q^\pi \right\|_\mu + \epsilon \left\| \theta^\pi \right\|_2 \sup_{s \in \mathcal{S}, a \in \mathcal{A}} \left\| \phi(s,a) \right\|_2,$$

*where $Q^\pi$ is the exact solution to Equation 1 and $\theta^\pi = \arg\min_\theta \left\| \Phi\theta - Q^\pi \right\|_\mu$.*

*Proof.* First note that Theorem 1 implies the convergence of SARSA(1) with tug-of-war hashing to a unique solution, which we denote $\tilde{\theta}^\pi$. We first apply Lemma 2 to the set $\{\phi(s,a) : (s,a) \in \mathcal{S} \times \mathcal{A}\}$ and $\theta^\pi$; note that we can safely assume $|\mathcal{S}||\mathcal{A}| > 1$, and therefore $\delta' < 1/10$. By our choice of $m$, for all $(s,a) \in \mathcal{S} \times \mathcal{A}$, $|H\phi(s,a) \cdot H\theta^\pi - \phi(s,a) \cdot \theta^\pi| \leq \epsilon \left\| \phi(s,a) \right\|_2 \left\| \theta^\pi \right\|_2$ with probability at least $1 - 6|\mathcal{S}||\mathcal{A}|\delta' = 1 - \delta$. As previously noted, SARSA(1) converges to $\tilde{\theta}^\pi = \arg\min_\theta \left\| \tilde{\Phi}\theta - Q^\pi \right\|_\mu$; compared to $\tilde{\Phi}\tilde{\theta}^\pi$, the solution $\theta_H^\pi := \tilde{\Phi}H\theta^\pi$ is thus an equal or worse approximation to $Q^\pi$. It follows that

$$
\begin{aligned}
\left\| \tilde{\Phi}\tilde{\theta}^\pi - Q^\pi \right\|_\mu &\leq \left\| \tilde{\Phi}\theta_H^\pi - Q^\pi \right\|_\mu \leq \left\| \tilde{\Phi}\theta_H^\pi - \Phi\theta^\pi \right\|_\mu + \left\| \Phi\theta^\pi - Q^\pi \right\|_\mu \\
&= \sqrt{\sum_{s \in \mathcal{S}, a \in \mathcal{A}} \mu(s,a) \left[ H\phi(s,a) \cdot H\theta^\pi - \phi(s,a) \cdot \theta^\pi \right]^2} + \left\| \Phi\theta^\pi - Q^\pi \right\|_\mu \\
&\leq \sqrt{\sum_{s \in \mathcal{S}, a \in \mathcal{A}} \mu(s,a) \left[ \epsilon \left\| \phi(s,a) \right\|_2 \left\| \theta^\pi \right\|_2 \right]^2} + \left\| \Phi\theta^\pi - Q^\pi \right\|_\mu \quad \text{(Lemma 2)} \\
&\leq \epsilon \left\| \theta^\pi \right\|_2 \sup_{s \in \mathcal{S}, a \in \mathcal{A}} \left\| \phi(s,a) \right\|_2 + \left\| \Phi\theta^\pi - Q^\pi \right\|_\mu,
\end{aligned}
$$

as desired. $\qquad\square$

Our proof of Theorem 2 critically requires the use of $\lambda = 1$. A natural next step would be to attempt to drop this restriction on $\lambda$. It also seems likely that the finite-sample analysis of LSTD with random projections [11] can be extended to cover the case of tug-of-war hashing. Theorem 2 suggests that, under the right conditions, the tug-of-war value function is a good approximation to the full-vector value function. A natural question now arises: does tug-of-war hashing lead to improved linear value function approximation compared with standard hashing? More importantly, does tug-of-war hashing result in better learned policies? These are the questions we investigate empirically in the next section.

## 4 Experimental Study

In the sketch setting, the appeal of tug-of-war hashing over standard hashing lies in its unbiasedness. We therefore begin with an empirical study of the magnitude of the bias when applying different hashing methods in a value function approximation setting.

### 4.1 Value Function Bias

We used standard hashing, tug-of-war hashing, and no hashing to learn a value function over a short trajectory in the Mountain Car domain [22]. Our evaluation uses a standard implementation available online [15].

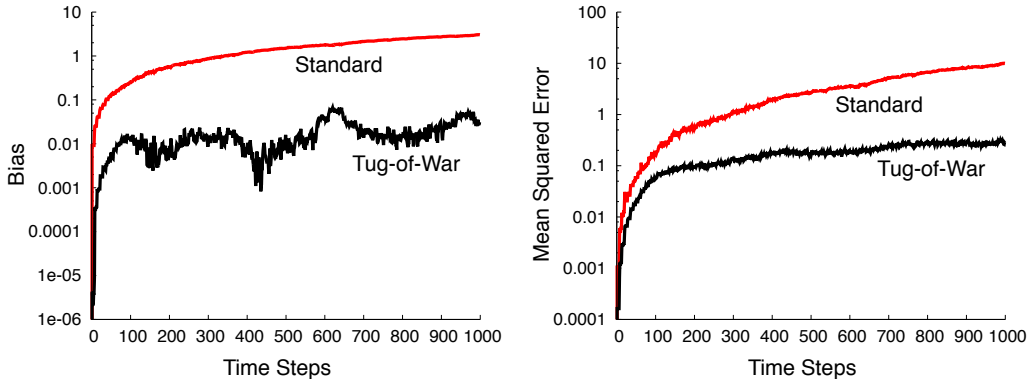

Figure 1: Bias and Mean Squared Error of value estimates using standard and tug-of-war hashing in 1,000 learning steps of Mountain Car. Note the log scale of the $y$ axis.

We generated a 1,000-step trajectory using an $\epsilon$-greedy policy [23]. For this fixed trajectory we updated a full feature weight vector $\theta_t$ using SARSA(0) with $\gamma = 1.0$ and $\alpha = 0.01$. We focus on SARSA(0) as it is commonly used in practice for its ease of implementation and its faster update speed in sparse settings. Parallel to the full-vector update we also updated both a tug-of-war weight vector $\tilde{\theta}_t$ and a standard hashing weight vector $\hat{\theta}_t$, with the same values of $\gamma$ and $\alpha$. Both methods use a hash table size of $m = 100$ and the same randomly selected hash function. This hash function is defined as $(ax + b) \mod p \mod m$, where $p$ is a large prime and $a, b < p$ are random integers [5]. At every step we compute the difference in value between the hashed value functions $\tilde{Q}_t(s_t, a_t)$ and $\hat{Q}_t(s_t, a_t)$, and the full-vector value function $Q_t(s_t, a_t)$. We repeated this experiment using 1 million hash functions selected uniformly at random. Figure 1 shows for each time step, estimates of the magnitude of the biases $\mathbb{E}[\tilde{Q}_t(s_t, a_t)] - Q_t(s_t, a_t)$ and $\mathbb{E}[\hat{Q}_t(s_t, a_t)] - Q_t(s_t, a_t)$ as well as estimates of the mean squared errors $\mathbb{E}[(\tilde{Q}_t(s_t, a_t) - Q_t(s_t, a_t))^2]$ and $\mathbb{E}[(\hat{Q}_t(s_t, a_t) - Q_t(s_t, a_t))^2]$ using the different hash functions. To provide a sense of scale, the estimate of the value of the final state when using no hashing is approximately $-4$; note that the $y$-axis uses a logarithmic scale.

The tug-of-war value function has a small, almost negligible bias. In comparison, the bias of standard hashing is orders of magnitude larger – almost as large as the value it is trying to estimate. The mean square error estimates show a similar trend. Furthermore, the same experiment on the Acrobot domain [22] yielded qualitatively similar results. Our results confirm the insights provided in Section 2: the tug-of-war value function can be significantly less biased than the standard hashing value function.

## 4.2 Reinforcement Learning Performance

Having smaller bias and mean square error in the $Q$-value estimates does not necessarily imply improved agent performance. In reinforcement learning, actions are selected based on relative $Q$-values, so a consistent bias may be harmless. In this section we evaluate the performance (cumulative reward per episode) of learning agents using both tug-of-war and standard hashing.

### 4.2.1 Tile Coding

We first studied the performance of agents using each of the two hashing methods in conjunction with tile coding. Our study is based on Mountain Car and Acrobot, two standard RL benchmark domains. For both domains we used the standard environment dynamics [22]; we used the fixed starting-state version of Mountain Car to reduce the variance in our results. We compared the two hashing methods using $\epsilon$-greedy policies and the SARSA($\lambda$) algorithm.

For each domain and each hashing method we performed a parameter sweep over the learning rate $\alpha$ and selected the best value which did not cause the value estimates to divergence. The Acrobot state was represented using 48 $6 \times 6 \times 6 \times 6$ tilings and the Mountain Car state, 10 $9 \times 9$ tilings. Other parameters were set to $\gamma = 1.0$, $\lambda = 0.9$, $\epsilon = 0.0$; the learning rate was further divided by the number of tilings.

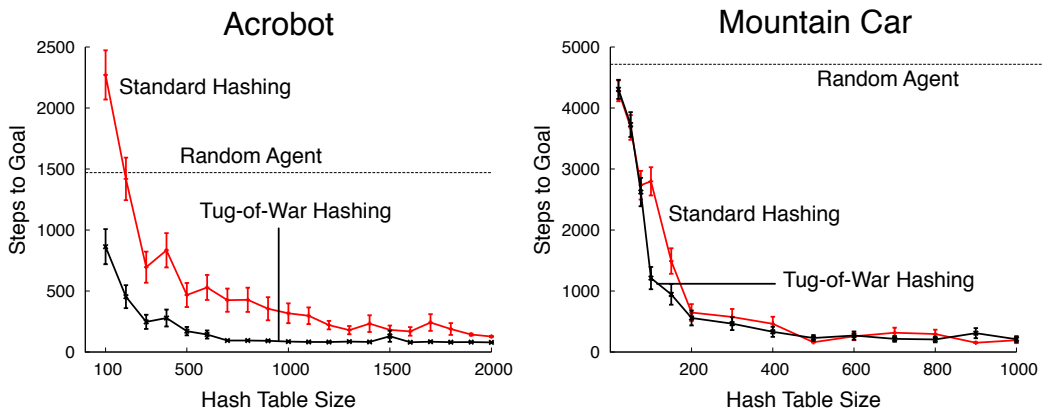

Figure 2: Performance of standard hashing and tug-of-war hashing in two benchmark domains. The performance of the random agent is provided as reference.

We experimented with hash table sizes $m \in [20, 1000]$ for Mountain Car and $m \in [100, 2000]$ for Acrobot. Each experiment consisted of 100 trials, sampling a new hash function for each trial. Each trial consisted of 10,000 episodes, and episodes were restricted to 5,000 steps. At the end of each trial, we disabled learning by setting $\alpha = 0$ and evaluated the agent on an additional 500 episodes.

Figure 2 shows the performance of standard hashing and tug-of-war hashing as a function of the hash table size. The conclusion is clear: when the hashed vector is small relative to the full vector, tug-of-war hashing performs better than standard hashing. This is especially true in Acrobot, where the number of features (over 62,000) necessarily results in harmful collisions.

### 4.2.2 Atari

We next evaluated tug-of-war hashing and standard hashing on a suite of Atari 2600 games. The Atari domain was proposed as a game-independent platform for AI research by Naddaf [17]. Atari games pose a variety of challenges for learning agents. The learning agent's observation space is the game screen: 160x210 pixels, each taking on one of 128 colors. In the game-independent setting, agents are tuned using a small number of training games and subsequently evaluated over a large number of games for which no game-specific tuning takes place. The game-independent setting forces us to use features that are common to all games, for example, by encoding the presence of color patterns in game screens; such an encoding is a form of exhaustive feature generation. Different learning methods have been evaluated on the Atari 2600 platform [9, 26, 12]. We based our evaluation on prior work on a suite of Atari 2600 games [2], to which we refer the reader for full details on handling Atari 2600 games as RL domains. We performed parameter sweeps over five training games, and tested our algorithms on fifty testing games.

We used models of contingency awareness to locate the player avatar [2]. From a given game, we generate feature sets by exhaustively enumerating all single-color patterns of size 1x1 (single pixels), 2x2, and 3x3. The presence of each different pattern within a 4x5 tile is encoded as a binary feature. We also encode the *relative* presence of patterns with respect to the player avatar location. This procedures gives rise to 569,856,000 different features, of which 5,000 to 15,000 are active at a given time step.

We trained $\epsilon$-greedy SARSA(0) agents using both standard hashing and tug-of-war hashing with hash tables of size m=1,000, 5,000 and 20,000. We chose the step-size $\alpha$ using a parameter sweep over the training games: we selected the best-performing $\alpha$ which never resulted in divergence in the value function. For standard hashing, $\alpha = 0.01, 0.05, 0.2$ for $m = 1,000, 5,000$ and $20,000$, respectively. For tug-of-war hashing, $\alpha = 0.5$ across table sizes. We set $\gamma = 0.999$ and $\epsilon = 0.05$. Each experiment was repeated over ten trials lasting 10,000 episodes each; we limited episodes to 18,000 frames to avoid issues with non-terminating policies.

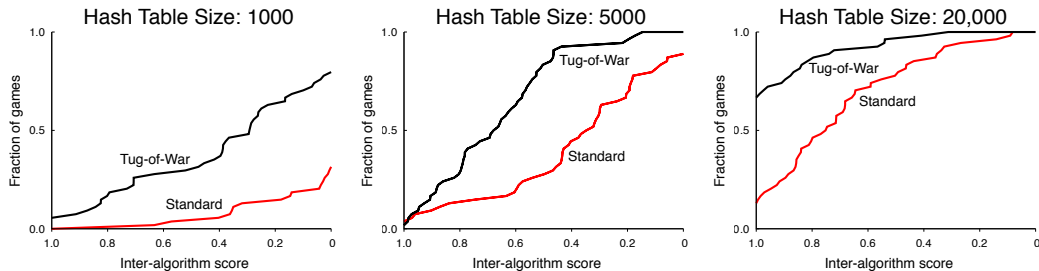

Figure 3: Inter-algorithm score distributions over fifty-five Atari games. Higher curves reflect higher normalized scores.

Accurately comparing methods across fifty-five games poses a challenge, as each game exhibits a different reward function and game dynamics. We compared methods using inter-algorithm score distributions [2]. For each game, we extracted the average score achieved by our agents over the last 500 episodes of training, yielding six different scores (three per hashing method) per game. Denoting these scores by $s_{g,i}$, $i = 1 \ldots 6$, we defined the inter-algorithm normalized score $z_{g,i} := (s_{g,i} - r_{g,\min})/(r_{g,\max} - r_{g,\min})$ with $r_{g,\min} := \min_i \{s_{g,i}\}$ and $r_{g,\max} := \max_i \{s_{g,i}\}$. Thus $z_{g,i} = 1.0$ indicates that the $i^{th}$ score was the highest for game $g$, and $z_{g,i} = 0.0$ similarly indicates the lowest score. For each combination of hashing method and memory size, its inter-algorithm score distribution shows the fraction of games for which the corresponding agent achieves a certain normalized score or better.

Figure 3 compares the score distributions of agents using either standard hashing or tug-of-war hashing for $m = 1,000$, 5,000 and 20,000. Tug-of-war hashing consistently outperforms standard hashing across hash table sizes. For each $m$ and each game, we also performed a two-tailed Welch's $t$-test with 99% confidence intervals to determine the statistical significance of the average score difference between the two methods. For $m = 1,000$, tug-of-war hashing performed statistically better in 38 games and worse in 5; for $m = 5,000$, it performed better in 41 games and worse in 7; and for $m = 20,000$ it performed better in 35 games and worse in 5. Our results on Atari games confirm what we observed on Mountain Car and Acrobot: in practice, tug-of-war hashing performs much better than standard hashing. Furthermore, computing the $\xi$ function took less than 0.3% of the total experiment time, a negligible cost in comparison to the benefits of using tug-of-war hashing.

## 5 Conclusion

In this paper, we cast the tug-of-war sketch into the reinforcement learning framework. We showed that, although the tug-of-war sketch is unbiased in the setting for which it was developed [7], the self-referential component of reinforcement learning induces a small bias. We showed that this bias can be much smaller than the bias that results from standard hashing and provided empirical results confirming the superiority of tug-of-war hashing for value function approximation.

As increasingly more complex reinforcement learning problems arise and strain against the boundaries of practicality, so the need for fast and reliable approximation methods grows. If standard hashing frees us from the curse of dimensionality, then tug-of-war hashing goes a step further by ensuring, when the demands of the task exceed available resources, a robust and principled shift from the exact solution to its approximation.

**Acknowledgements**

We would like to thank Bernardo Ávila Pires, Martha White, Yasin Abbasi-Yadkori and Csaba Szepesvári for the help they provided with the theoretical aspects of this paper, as well as Adam White and Rich Sutton for insightful discussions on hashing and tile coding. This research was supported by the Alberta Innovates Technology Futures and the Alberta Innovates Centre for Machine Learning at the University of Alberta. Invaluable computational resources were provided by Compute/Calcul Canada.

## Footnotes

[1]While it may seem odd to randomly select your hash function, this can equivalently be thought as sampling an indexing assignment for the MDP's features. While a particular hash function may be well- (or poorly-) suited for a particular MDP, it is hard to imagine how this could be known a priori. By considering a randomly selected hash function (or random permutation of the features), we are simulating the uncertainty of using a particular hash function on a never before encountered MDP.

# References

[1] Dimitris Achlioptas. Database-friendly random projections: Johnson-Lindenstrauss with binary coins. *Journal of Computer and System Sciences*, 66(4):671–687, 2003.

[2] Marc G. Bellemare, Joel Veness, and Michael Bowling. Investigating contingency awareness using Atari 2600 games. In *Proceedings of the Twenty-Sixth AAAI Conference on Artificial Intelligence*, 2012.

[3] Michael Bowling and Manuela Veloso. Scalable learning in stochastic games. In *AAAI Workshop on Game Theoretic and Decision Theoretic Agents*, 2002.

[4] Michael Bowling and Manuela Veloso. Simultaneous adversarial multi-robot learning. In *Proceedings of the Eighteenth International Joint Conference on Artificial Intelligence*, pages 699–704, 2003.

[5] J. Lawrence Carter and Mark N. Wegman. Universal classes of hash functions. *Journal of Computer and System Sciences*, 18(2):143–154, 1979.

[6] Graham Cormode. Sketch techniques for massive data. In Graham Cormode, Minos Garofalakis, Peter Haas, and Chris Jermaine, editors, *Synopses for Massive Data: Samples, Histograms, Wavelets and Sketches*, Foundations and Trends in Databases. NOW publishers, 2011.

[7] Graham Cormode and Minos Garofalakis. Sketching streams through the net: Distributed approximate query tracking. In *Proceedings of the 31st International Conference on Very Large Data Bases*, pages 13–24, 2005.

[8] Anirban Dasgupta, Ravi Kumar, and Tamás Sarlós. A sparse Johnson-Lindenstrauss transform. In *Proceedings of the 42nd ACM Symposium on Theory of Computing*, pages 341–350, 2010.

[9] Carlos Diuk, A. Andre Cohen, and Michael L. Littman. An object-oriented representation for efficient reinforcement learning. In *Proceedings of the Twenty-Fifth International Conference on Machine Learning*, pages 240–247, 2008.

[10] Mahdi Milani Fard, Yuri Grinberg, Joelle Pineau, and Doina Precup. Compressed least-squares regression on sparse spaces. In *Proceedings of the Twenty-Sixth AAAI Conference on Artificial Intelligence*. AAAI, 2012.

[11] Mohammad Ghavamzadeh, Alessandro Lazaric, Oldaric-Ambrym Maillard, and Rémi Munos. LSTD with random projections. In *Advances in Neural Information Processing Systems 23*, pages 721–729, 2010.

[12] Matthew Hausknecht, Piyush Khandelwal, Risto Miikkulainen, and Peter Stone. HyperNEAT-GGP: A HyperNEAT-based Atari general game player. In *Genetic and Evolutionary Computation Conference (GECCO)*, 2012.

[13] Daniel M. Kane and Jelani Nelson. A derandomized sparse Johnson-Lindenstrauss transform. *arXiv preprint arXiv:1006.3585*, 2010.

[14] Ping Li, Trevor J. Hastie, and Kenneth W. Church. Very sparse random projections. In *Proceedings of the 12th ACM SIGKDD International Conference on Knowledge Discovery and Data Mining*, pages 287–296, 2006.

[15] The Reinforcement Learning Library, 2010. http://library.rl-community.org.

[16] Oldaric-Ambrym Maillard and Rémi Munos. Compressed least squares regression. In *Advances in Neural Information Processing Systems 22*, pages 1213–1221, 2009.

[17] Yavar Naddaf. *Game-independent AI agents for playing Atari 2600 console games*. PhD thesis, University of Alberta, 2010.

[18] E. Schuitema, D.G.E. Hobbelen, P.P. Jonker, M. Wisse, and J.G.D. Karssen. Using a controller based on reinforcement learning for a passive dynamic walking robot. In *Proceedings of the Fifth IEEE-RAS International Conference on Humanoid Robots*, pages 232–237, 2005.

[19] David Silver, Richard S. Sutton, and Martin Müller. Reinforcement learning of local shape in the game of Go. In *20th International Joint Conference on Artificial Intelligence*, pages 1053–1058, 2007.

[20] Peter Stone, Richard S. Sutton, and Gregory Kuhlmann. Reinforcement learning for RoboCup soccer keepaway. *Adaptive Behavior*, 13(3):165, 2005.

[21] Nathan Sturtevant and Adam White. Feature construction for reinforcement learning in Hearts. *Computers and Games*, pages 122–134, 2006.

[22] Richard S. Sutton. Generalization in reinforcement learning: Successful examples using sparse coarse coding. In David S. Touretzky, Michael C. Mozer, and Michael E. Hasselmo, editors, *Advances in Neural Information Processing Systems*, volume 8, pages 1038–1044, 1996.

[23] Richard S. Sutton and Andrew G. Barto. *Reinforcement Learning: An Introduction*. MIT Press, 1998.

[24] Russ Tedrake, Teresa Weirui Zhang, and H. Sebastian Seung. Stochastic policy gradient reinforcement learning on a simple 3D biped. In *Proceedings of Intelligent Robots and Systems 2004*, volume 3, pages 2849–2854, 2004.

[25] John N. Tsitsiklis and Benjamin Van Roy. An analysis of temporal-difference learning with function approximation. *IEEE Transactions on Automatic Control*, 42(5):674–690, 1997.

[26] Samuel Wintermute. Using imagery to simplify perceptual abstraction in reinforcement learning agents. In *Proceedings of the Twenty-Fourth AAAI Conference on Artificial Intelligence*, 2010.

